# Sparse Representation for Gaussian Process Models

**Lehel Csató    and    Manfred Opper**
Neural Computing Research Group
School of Engineering and Applied Sciences
B4 7ET Birmingham, United Kingdom
{csatol,opperm}@aston.ac.uk

## Abstract

We develop an approach for a sparse representation for Gaussian Process (GP) models in order to overcome the limitations of GPs caused by large data sets. The method is based on a combination of a Bayesian online algorithm together with a sequential construction of a relevant subsample of the data which fully specifies the prediction of the model. Experimental results on toy examples and large real-world datasets indicate the efficiency of the approach.

## 1  Introduction

Gaussian processes (GP) [1; 15] provide promising non-parametric tools for modelling real-world statistical problems. Like other kernel based methods, e.g. Support Vector Machines (SVMs) [13], they combine a high flexibility of the model by working in high (often $\infty$) dimensional feature spaces with the simplicity that all operations are "kernelized" i.e. they are performed in the (lower dimensional) input space using positive definite kernels.

An important advantage of GPs over other non-Bayesian models is the explicit probabilistic formulation of the model. This does not only provide the modeller with (Bayesian) confidence intervals (for regression) or posterior class probabilities (for classification) but also immediately opens the possibility to treat other nonstandard data models (e.g. Quantum inverse statistics [4]).

Unfortunately the drawback of GP models (which was originally apparent in SVMs as well, but has now been overcome [6]) lies in the huge increase of the computational cost with the number of training data. This seems to preclude applications of GPs to large datasets.

This paper presents an approach to overcome this problem. It is based on a combination of an online learning approach requiring only a single sweep through the data and a method to reduce the number of parameters representing the model.

Making use of the proposed parametrisation the method extracts a subset of the examples and the prediction relies only on these *basis vectors* (BV). The memory requirement of the algorithm scales thus only with the size of this set. Experiments with real-world datasets confirm the good performance of the proposed method. [1]

## 2 Gaussian Process Models

GPs belong to Bayesian non-parametric models where likelihoods are parametrised by a Gaussian stochastic process (random field) $a(\boldsymbol{x})$ which is indexed by the continuous input variable $\boldsymbol{x}$. The prior knowledge about $a$ is expressed in the prior mean and the covariance given by the kernel $K_0(\boldsymbol{x}, \boldsymbol{x}') = Cov(a(\boldsymbol{x}), a(\boldsymbol{x}'))$ [14; 15]. In the following, only zero mean GP priors are used.

In supervised learning the process $a(\boldsymbol{x})$ is used as a latent variable in the likelihood $P(y|a(\boldsymbol{x}))$ which denotes the probability of output $y$ given the input $\boldsymbol{x}$. Based on a set of input-output pairs $(\boldsymbol{x}_n, y_n)$ with $\boldsymbol{x}_n \in R^m$ and $y_n \in R$ $(n = 1, N)$ the Bayesian learning method computes the posterior distribution of the process $a(\boldsymbol{x})$ using the prior and likelihood [14; 15; 3].

Although the prior is a Gaussian process, the posterior process usually is not Gaussian (except for the special case of regression with Gaussian noise). Nevertheless, various approaches have been introduced recently to approximate the posterior averages [11; 9]. Our approach is based on the idea of approximating the true posterior process $p\{a\}$ by a Gaussian process $q\{a\}$ which is fully specified by a covariance kernel $K_t(\boldsymbol{x}, \boldsymbol{x}')$ and posterior mean $\langle a(\boldsymbol{x})\rangle_t$, where $t$ is the number of training data processed by the algorithm so far. Such an approximation could be formulated within the variational approach, where $q$ is chosen such that the relative entropy $D(q, p) \doteq \mathrm{E}_q \ln \frac{dq}{dp}$ is minimal [9]. However, in this formulation, the expectation is over the approximate process $q$ rather than over $p$. It seems intuitively better to minimise the other KL divergence given by $D(p, q) \doteq \mathrm{E}_p \ln \frac{dp}{dq}$, because the expectation is over the true distribution. Unfortunately, such a computation is generally not possible. The following online approach can be understood as an approximation to this task.

## 3 Online learning for Gaussian Processes

In this section we briefly review the main idea of the Bayesian online approach (see e.g. [5]) to GP models. We process the training data sequentially one after the other. Assume we have a Gaussian approximation to the posterior process at time $t$. We use the next example $t + 1$ to update the posterior using Bayes rule via

$$\tilde{p}(\boldsymbol{a}) = \frac{P(y_{t+1}|a(\boldsymbol{x}_{t+1}))p_t(\boldsymbol{a})}{\langle P(y_{t+1}|a(\boldsymbol{x}_{t+1}))\rangle_t}$$

Since the resulting posterior $\tilde{p}(\boldsymbol{a})$ is non-Gaussian, we project it to the closest Gaussian process $q$ which minimises the KL divergence $D(\tilde{p}, q)$. Note, that now the new approximation $q$ is on "correct" side of the KL divergence. The minimisation can be performed exactly, leading to a match of the means and covariances of $\tilde{p}$ and $q$. Since $\tilde{p}$ is much less complex than the full posterior, it is possible to write down the changes in the first two moments analytically [2]:

$$\begin{aligned}\langle a(\boldsymbol{x})\rangle_{t+1} &= \langle a(\boldsymbol{x})\rangle_t + b_1 \, K_t(\boldsymbol{x}, \boldsymbol{x}_{t+1}) \\ K_{t+1}(\boldsymbol{x}, \boldsymbol{x}') &= K_t(\boldsymbol{x}, \boldsymbol{x}') + b_2 K_t(\boldsymbol{x}, \boldsymbol{x}_{t+1}) K_t(\boldsymbol{x}_{t+1}, \boldsymbol{x}')\end{aligned} \quad (1)$$

where the scalar coefficients $b_1$ and $b_2$ are:

$$b_1 = \partial_{a_t} \ln\langle P(y_{t+1}|a(\boldsymbol{x}_{t+1}))\rangle_t \qquad b_2 = \partial_{a_t}^2 \ln\langle P(y_{t+1}|a(\boldsymbol{x}_{t+1}))\rangle_t \quad (2)$$

with averaging performed with respect to the *marginal* Gaussian distribution of the process variable $a$ at input $\boldsymbol{x}_{t+1}$. Note, that this yields *a one dimensional* integral! Derivatives are

---

is based on splitting the data-set into smaller subsets and training individual GP predictors on each of them. The final prediction is achieved by a specific weighting of the individual predictors.

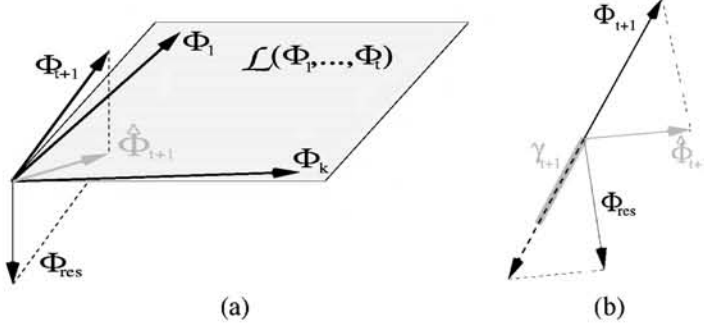

<div align="center">(a)        (b)</div>

Figure 1: Projection of the new input $\Phi_{t+1}$ to the subspace spanned by previous inputs. $\hat{\Phi}_{t+1}$ is the projection to the linear span of $\{\Phi_i\}_{i=1,t}$, and $\Phi_{res}$ the residual vector. Subfigure (a) shows the projections to the subspace, and (b) gives a geometric picture of the "measurable part" of the error $\gamma_{t+1}$ from eq. (8).

taken with respect to $\langle a(\boldsymbol{x}) \rangle_t$. Note also that this procedure does *not* equal the extended Kalman filter which involves *linearisations* of likelihoods, whereas in our approach it is possible to use non-smooth likelihoods (e.g. noise free classifications) without problems.

It turns out, that the recursion (1) is solved by the parametrisation

$$
\begin{aligned}
\langle a(\boldsymbol{x}) \rangle_t &= \sum_{i=1}^{t} K_0(\boldsymbol{x}, \boldsymbol{x}_i) \alpha_t(i) \\
K_t(\boldsymbol{x}, \boldsymbol{x}') &= K_0(\boldsymbol{x}, \boldsymbol{x}') + \sum_{i,j=1}^{t} K_0(\boldsymbol{x}, \boldsymbol{x}_i) C_t(ij) K_0(\boldsymbol{x}_j, \boldsymbol{x}')
\end{aligned}
\tag{3}
$$

such that in each on-line step, we have to update only the vector of $\alpha$'s and the matrix of $C$'s. For notational convenience we use vector $\boldsymbol{\alpha}_t = [\alpha_t(1), \ldots, \alpha_t(N)]^T$ and matrix $C_t = \{C_t(ij)\}_{i,j=1,N}$. Zero-mean GP with kernel $K_0$ is used as the starting point for the algorithm: $\boldsymbol{\alpha}_0 = 0$ and $C_0 = 0$ will be the starting parameters.

The update of the parameters defined in (3) is found to be

$$
\begin{aligned}
\boldsymbol{\alpha}_{t+1} &= \boldsymbol{\alpha}_t + b_1 \left[ C_t \boldsymbol{k}_{t+1} + \boldsymbol{e}_{t+1} \right] \\
C_{t+1} &= C_t + b_2 \left[ C_t \boldsymbol{k}_{t+1} + \boldsymbol{e}_{t+1} \right] \left[ C_t \boldsymbol{k}_{t+1} + \boldsymbol{e}_{t+1} \right]^T
\end{aligned}
\tag{4}
$$

with $\boldsymbol{k}_{t+1} = [K_0(\boldsymbol{x}_1, \boldsymbol{x}_{t+1}), \ldots, K_0(\boldsymbol{x}_t, \boldsymbol{x}_{t+1})]^T$, $\boldsymbol{e}_{t+1}$ the $t+1$-th unit vector (all components except $t+1$-th are zero), and the scalar coefficients $b_1$ and $b_2$ computed from (2).

The serious drawback of this approach, which it shares with many other kernel methods, is the quadratic increase of the matrix size with the training data.

## 4 Sparse representation

We use the following idea for reducing the increase of the size of $C$ and $\alpha$ (for a similar approach see [8]). We consider the feature expansion of the kernel $K_0(\boldsymbol{x}, \boldsymbol{x}') = \Phi(\boldsymbol{x})^T \Phi(\boldsymbol{x}')$ and decompose the new feature vector $\Phi(\boldsymbol{x}_{t+1})$ as a linear combination of the previous features and a residual $\Phi_{res}$:

$$
\Phi(\boldsymbol{x}_{t+1}) = \hat{\Phi}_{t+1} + \Phi_{res} = \sum_{i=1}^{t} \hat{e}_i \Phi(\boldsymbol{x}_i) + \Phi_{res}
\tag{5}
$$

where $\hat{\Phi}_{t+1}$ is the projection of $\Phi_{t+1}$ to the previous inputs and $\hat{\boldsymbol{e}}_{t+1} = [\hat{e}_1, \ldots, \hat{e}_t]^T$ are the coordinates of $\hat{\Phi}_{t+1}$ with respect to the basis $\{\Phi_i\}_{i=1,t}$. We can then re-express the GP means:

$$
\langle a(\boldsymbol{x}) \rangle_{t+1} = \sum_{i=1}^{t} \hat{\alpha}_{t+1}(i) \Phi(\boldsymbol{x}_i)^T \Phi(\boldsymbol{x}) + \gamma_{t+1} \Phi_{res}^T \Phi(\boldsymbol{x})
\tag{6}
$$

with $\hat{\alpha}_{t+1}(i) = \alpha_{t+1}(i) + \hat{e}_{t+1}(i)\alpha_{t+1}(t+1)$ and $\gamma_{t+1}$ the residual (or novelty factor) associated with the new feature vector. The vector $\hat{e}_{t+1}$ and the residual term $\gamma_{t+1}$ are all expressed in terms of kernels:

$$\hat{e}_{t+1} = K_B^{(-1)} k_{t+1} \qquad \gamma_{t+1} = k_{t+1}^* - k_{t+1}^T K_B^{(-1)} k_{t+1} \qquad (7)$$

with $K_B(ij) = \{K_0(x_i, x_j)\}_{i,j=1,t}$ and $k_{t+1}^* = K_0(x_{t+1}, x_{t+1})$. The relation between the quantities $\hat{e}_{t+1}$ and $\gamma_{t+1}$ is illustrated in Figure 1.

Neglecting the last term in the decomposition of the new input (5) and performing the update with the resulting vector is equivalent to the update rule (4) with $e_{t+1}$ replaced by $\hat{e}_{t+1}$. Note that the dimension of parameter space *is not increased* by this approximative update. The memory required by the algorithm scales quadratically only with the size of the set of "basis vectors", i.e. those examples for which the full update (4) is made. This is similar to Support Vectors [13], without the need to solve the (high dimensional) convex optimisation problem. It is also related to the kernel PCA and the reduced set method [8] where the full solution is computed first and then a reduced set is used for prediction.

Replacing the input vector $\Phi_{t+1}$ by its projection on the linear span of the BVs when updating the GP parameters induces changes in the GP[2]. However, the replacement of the true feature vector by its approximation leaves the mean function unchanged at each BV $i = 1, t$. That is, the functions $\langle a(x) \rangle_{t+1}$ from (6) and $\langle \hat{a}(x) \rangle_{t+1} = \sum_{i=1}^{t} \hat{\alpha}_{t+1}(i) K_0(x_i, x)$ have the same value at all $x_l$. The change at $x_{t+1}$ is

$$\varepsilon_{t+1} = |\langle a(x_{t+1}) \rangle_{t+1} - \langle \hat{a}(x_{t+1}) \rangle_{t+1}| = |b_1| \gamma_{t+1} \qquad (8)$$

with $b_1$ the factor from (2).

As a consequence, a good approximation to the full GP solution is obtained if the input for which we have only a small change in the mean function of the posterior process is not included in the set of BVs. The change is given by $\varepsilon_{t+1}$ and the decision of including $x_{t+1}$ or not is based on the *"score"* associated to it.

The absence of matrix inversions is an important issue when dealing with large datasets. The matrix inversion from the projection equation (7) can be avoided by iterative inversion[3] of the Gram matrix $Q = K_B^{-1}$:

$$Q_{t+1} = Q_t + \gamma_{t+1}^{-1} (\hat{e}_{t+1} - e_{t+1})(\hat{e}_{t+1} - e_{t+1})^T \qquad (9)$$

An important comment is that if the new input is in the linear span of the BVs, then it will not be included in the basis set, avoiding thus: 1.) the small singular values of the matrix $K_B$ and 2.) the redundancy in representing the problem.

### 4.1 Deleting a basis vector

The above section gave a method to leave out a vector that is not significant for the prediction purposes. However, it did not provide us with a method to eliminate one of the already existing BV-s.

Let us assume that an input $x_{t+1}$ has just been added to the set of BVs. Since we know that an addition had taken place, the update rule (4) with the $t + 1$-th unit vector $e_{t+1}$ was last performed. Since the model parameters at the previous step had an *empty* $t + 1$-th row and column, the parameters before the *full* update can be identified.

The removal of the last basis vector can be done with the following steps: 1) computing the parameters before the update of the GP and 2) performing a reduced update of the

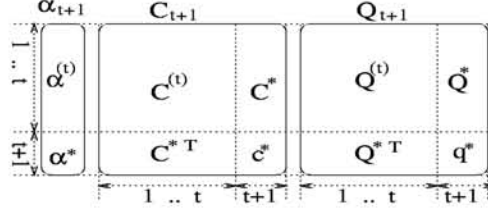

Figure 2: Decomposition of model parameters for the update equation (10).

model without the inclusion of the basis vector (eq. (4) using $\hat{e}_{t+1}$). The updates for model parameters $\alpha$, $C$, and $Q$ are "inverted" by inverting the coupled equations (4) and (9):

$$\hat{\alpha} = \alpha^{(t)} - \alpha^* \frac{Q^*}{q^*}$$

$$\hat{C} = C^{(t)} + c^* \frac{Q^* Q^{*T}}{q^{*2}} - \frac{1}{q^*} \left[ Q^* C^{*T} + C^* Q^{*T} \right] \qquad (10)$$

$$\hat{Q} = Q^{(t)} - \frac{Q^* Q^{*T}}{q*}$$

where the elements needed to update the model are extracted from the extended parameters as illustrated in Figure 2.

The consequence of the identification permits us to evaluate the score for the last BV. But since the order of the BVs is approximately arbitrary, we can assign a score to each BV

$$\varepsilon_i = \frac{|\alpha_{t+1}(i)|}{Q_{t+1}(i,i)}. \qquad (11)$$

Thus we have a method to estimate the score of each basis vector at any time and to eliminate the one with the least contribution to the GP output (the mean), providing a sparse GP with a full control over memory size.

## 5 Simulation results

To apply the online learning rules (4), the data likelihood for the specific problem has to be averaged with respect to a Gaussian. Using eq. (2), the coefficients $b_1$ and $b_2$ are obtained.

The marginal of the GP at $x_{t+1}$ is a normal distribution with mean $\langle a(x_{t+1}) \rangle_t = \alpha_t^T k_{t+1}$ and variance $\sigma^2_{x_{t+1}} = k^*_{t+1} + k^T_{t+1} C_t k_{t+1}$ where the GP parameters at time $t$ are considered. As a first example, we consider regression with Gaussian output noise $\sigma_0^2$ for which

$$\ln \langle P(y_{t+1}|a(x_{t+1})) \rangle_t = -\frac{1}{2} \ln \left( 2\pi(\sigma_0^2 + \sigma^2_{x_{t+1}}) \right) - \frac{(y_{t+1} - \langle a(x_{t+1}) \rangle_t)^2}{2(\sigma_0^2 + \sigma^2_{x_{t+1}})^2}. \qquad (12)$$

For classification we use the probit model. The outputs are binary $y \in \{-1, 1\}$ and the probability is given by the error function (where $u = ya/\sigma_0$):

$$P(y|a) = \mathrm{Erf}\left( \frac{ya}{\sigma_0} \right) = \frac{1}{\sqrt{2\pi}} \int_\infty^u dt e^{-t^2/2}$$

The averaged log-likelihood for the new data point at time $t$ is:

$$\langle P(y_{t+1}|a(x_{t+1})) \rangle = \mathrm{Erf}\left( \frac{y_{t+1} \, \alpha_t^T k_{t+1}}{\sqrt{\sigma_0^2 + \sigma^2_{x_{t+1}}}} \right) \qquad (13)$$

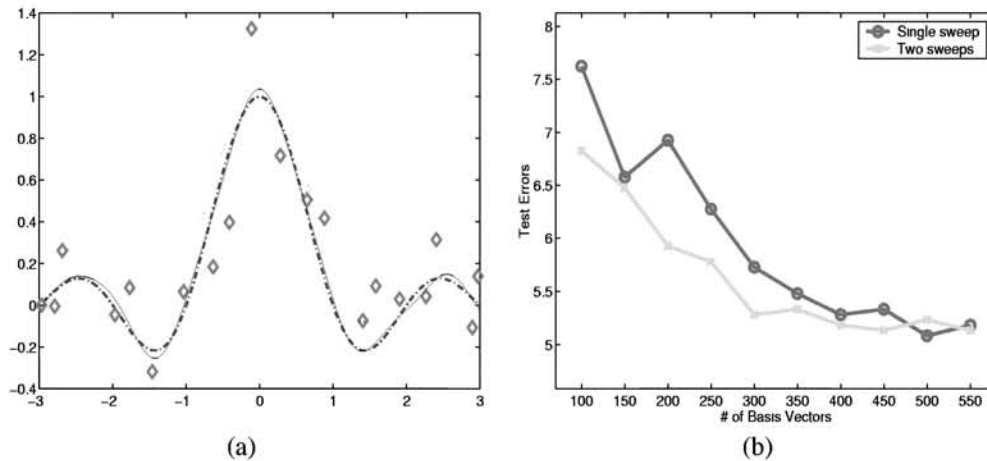

<div align="center">(a)&emsp;&emsp;&emsp;&emsp;&emsp;&emsp;&emsp;(b)</div>

Figure 3: Simulation results for regression (a) and classification (b). For details see text.

For the regression case we have chosen the toy data model $y = \sin(x)/x + \zeta$ where $\zeta$ is a zero-mean Gaussian random variable with variance $\sigma_0^2$ and an RBF kernel. Figure 3.a shows the result of applying the algorithm for 600 input data and restricting the number of BVs to 20. The dash-dotted line is the true function, the continuous line is the approximation with the Bayesian standard deviation plotted by dotted lines (a gradient-like approximation for the output noise based on maximising the likelihood (12) lead us to the variance with which the data has been generated).

For classification we used the data from the US postal database[4] of handwritten zip codes together with an RBF kernel. The database has 7291 training and 2007 test data of $16 \times 16$ grey-scale images. To apply the classification method to this database, 10 binary classification problems were solved and the final output was the class with the largest probability. The same BVs have been considered for each classifier and if a deletion was required, the BV having the minimum cumulative score was deleted. The cumulative score was chosen to be the maximum of the scores for each classifier. Figure 3.b shows the test error as a function of the size of the basis set. We find that the test error is rather stable over a considerable range of basis set sizes. Also a comparison with a second sweep through the data shows that the algorithm seems to have already extracted the relevant information out of the data within a single sweep. Using a polynomial kernel for the USPS dataset and 500 BVs we achieved a test error of 4.83%, which compares favourably with other sparse approaches [10; 8] but uses smaller basis sets than the SVM (2540 reported in [8]).

We also applied our algorithm to the NIST dataset[5] which contains 60000 data. Using a fourth order polynomial kernel with only 500 BVs we achieved a test error of 3.13% and we expect that improvements are possible by using a kernel with tunable hyperparameters. The possibility of computing the posterior class probabilities allows us to reject data. When the test data for which the maximum probability was below 0.5 was rejected, the test error was 1.53% with 1.60% of rejection rate.

## 6  Conclusion and further research

This paper presents a sparse approximation for GPs similar to the one found in SVMs [13] or relevance vector machines [10]. In contrast to these other approaches our algorithm is fully online and does not construct the sparse representation from the full data set (for sequential optimisation for SVM see [6]).

An important open question (besides the issue of model selection) is how to choose the minimal size of the set of basis vectors such that the predictive performance is not much deteriorated by the approximation involved. In fact, our numerical classification experiments suggest that the prediction performance is considerably stable when the basis set is above a certain size. It would be interesting if one could relate this minimum size to the effective dimensionality of the problem being defined as the number of feature dimensions which are well estimated by the data. One may argue as follows: Replacing the true kernel by a modified (finite dimensional) one which contains only the well estimated features will not change the predictive power. On the other hand, for kernels with a feature space of finite dimensionality $M$, it is easy to see that we need never more than $M$ basis vectors, because of linear dependence. Whether such reasoning will lead to a practical procedure for choosing the appropriate basis set size, is a question for further research.

## 7  Acknowledgement

This work was supported by EPSRC grant no. GR/M81608.

## Footnotes

[1] A different approach for dealing with large datasets was suggested by V. Tresp [12]. His method

[2]Equation (7) also minimises the KL-distance between the *full* posterior (the one that increases parameter space) and a parametric distribution using only the old BVs.

[3]A guide is available from Sam Roweis: http://www.gatsby.ucl.ac.uk/~roweis/notes.html

[4]From: http://www.kernel-machines.org/data.html

[5]Available from: http://www.research.att.com/~yann/ocr/mnist/

## References

[1] J. M. Bernardo and A. F. Smith. *Bayesian Theory*. John Wiley & Sons, 1994.

[2] L. Csató, E. Fokoué, M. Opper, B. Schottky, and O. Winther. Efficient approaches to Gaussian process classification. In *NIPS*, volume 12, pages 251–257. The MIT Press, 2000.

[3] M. Gibbs and D. J. MacKay. Efficient implementation of Gaussian processes. Technical report, http://wol.ra.phy.cam.ac.uk/mackay/abstracts/gpros.html, 1999.

[4] J. C. Lemm, J. Uhlig, and A. Weiguny. A Bayesian approach to inverse quantum statistics. *Phys.Rev.Lett.*, 84:2006, 2000.

[5] M. Opper. A Bayesian approach to online learning. In Saad [7], pages 363–378.

[6] J. C. Platt. Fast training of Support Vector Machines using sequential minimal optimisation. In *Advances in Kernel Methods (Support Vector Learning)*.

[7] D. Saad, editor. *On-Line Learning in Neural Networks*. Cambridge Univ. Press, 1998.

[8] B. Schölkopf, S. Mika, C. J. Burges, P. Knirsch, K.-R. Müller, G. Rätsch, and A. J. Smola. Input space vs. feature space in kernel-based methods. *IEEE Transactions on Neural Networks*, 10(5):1000–1017, September 1999.

[9] M. Seeger. Bayesian model selection for Support Vector Machines, Gaussian processes and other kernel classifiers. In S. A. Solla, T. K. Leen, and K.-R. Müller, editors, *NIPS*, volume 12. The MIT Press, 2000.

[10] M. Tipping. The Relevance Vector Machine. In S. A. Solla, T. K. Leen, and K.-R. Müller, editors, *NIPS*, volume 12. The MIT Press, 2000.

[11] G. F. Trecate, C. K. I. Williams, and M. Opper. Finite-dimensional approximation of Gaussian processes. In M. S. Kearns, S. A. Solla, and D. A. Cohn, editors, *NIPS*, volume 11. The MIT Press, 1999.

[12] V. Tresp. A Bayesian committee machine. *Neural Computation*, accepted.

[13] V. N. Vapnik. *The Nature of Statistical Learning Theory*. Springer-Verlag, New York, NY, 1995.

[14] C. K. I. Williams. Prediction with Gaussian processes. In M. I. Jordan, editor, *Learning in Graphical Models*. The MIT Press, 1999.

[15] C. K. I. Williams and C. E. Rasmussen. Gaussian processes for regression. In D. S. Touretzky, M. C. Mozer, and M. E. Hasselmo, editors, *NIPS*, volume 8. The MIT Press, 1996.
